# Explaining Away in Weight Space

**Peter Dayan**     **Sham Kakade**
Gatsby Computational Neuroscience Unit, UCL
17 Queen Square London WC1N 3AR
dayan@gatsby.ucl.ac.uk     sham@gatsby.ucl.ac.uk

## Abstract

Explaining away has mostly been considered in terms of inference of states in belief networks. We show how it can also arise in a Bayesian context in inference about the weights governing relationships such as those between stimuli and reinforcers in conditioning experiments such as *backward blocking*. We show how explaining away in weight space can be accounted for using an extension of a Kalman filter model; provide a new approximate way of looking at the Kalman gain matrix as a whitener for the correlation matrix of the observation process; suggest a network implementation of this whitener using an architecture due to Goodall; and show that the resulting model exhibits backward blocking.

## 1 Introduction

The phenomenon of explaining away is commonplace in inference in belief networks. In this, an explanation (a setting of activities of unobserved units) that is consistent with certain observations is accorded a low posterior probability if another explanation for the same observations is favoured either by the prior or by other observations. Explaining away is typically realized by recurrent inference procedures, such as mean field inference (see Jordan, 1998).

However, explaining away is not only important in the space of on-line explanations for data; it is also important in the space of *weights*. This is a very general problem that we illustrate using a phenomenon from animal conditioning called *backward blocking* (Shanks, 1985; Miller & Matute, 1996). Conditioning paradigms are important because they provide a window onto processes of successful natural inference, which are frequently statistically normative. Backwards blocking poses a very different problem from standard explaining away, and rather complex theories have been advanced to account for it (*eg* Wagner & Brandon, 1989). We treat it as a case for *Kalman filtering,* and suggest a novel network model for Kalman filtering to solve it. Consider three different conditioning paradigms associated with backwards blocking:

| name | set 1 | set 2 | test |
|---|---|---|---|
| forward | L→R | L,S→R | S→ • |
| backward | L,S→R | L→R | S→ • |
| sharing | L,S→R | —— | S→R/2 |

These paradigms involve one or two sets of multiple learning trials (set 1 and set 2), in which stimuli (a light, L, and/or a sound, S) are conditioned to a reward (R), followed by a

test phase, in which the strength of association between the sound S and reward is assessed. This is found to be weak (●) in forward and backward blocking, but stronger (R/2) in the sharing paradigm. The effect that concerns this paper is occurring during the second set of trials during backward blocking in which the association between the sound and the reward is weakened (compared with sharing), even though the sound is not presented during these trials. The apparent association between the sound and the reward established in the first set of trials is *explained away* in the second set of trials.

The standard explanation for this (Wagner's SOP model, see Wagner & Brandon, 1989) suggests that during the first set of trials, the light comes to predict the presence of the sound; and that during the second set of trials, the fact that the sound is expected (on the basis of the light, represented by the activation of 'opponent' sound units) but not presented, weakens the association between the sound and the reward. Not only does this suggestion lack a statistical basis, but also its network implementation requires that the activation of the opponent sound units makes weaker the weights from the standard sound units to reward. It is unclear how this could work.

In this paper, we first extend the Kalman filter based conditioning theory of Sutton (1992) to the case of backward blocking. Next, we show the close relationship between the key quantity for a Kalman filter — namely the covariance matrix of uncertainty about the relationship between the stimuli and the reward — and the symmetric whitening matrix for the stimuli. Then we show how the Goodall algorithm for whitening (Goodall 1960; Atick & Redlich, 1993) makes for an appropriate network implementation for weight updates based on the Kalman filter. The final algorithm is a motivated mixture of unsupervised and reinforcement (or, equivalently in this case, supervised) learning. Last, we demonstrate backward blocking in the full model.

## 2   The Kalman filter and classical conditioning

Sutton (1992) suggested that one can understand classical conditioning in terms of normative statistical inference. The idea is that on trial $n$ there is a set of true weights $\mathbf{w}_n$ mediating the relationship between the presentation of stimuli $\mathbf{x}_n$ and the amount of reward $r_n$ that is delivered, where

$$r_n = \mathbf{w}_n \cdot \mathbf{x}_n + \epsilon_n \tag{1}$$

and $\epsilon_n \sim N[0, \tau^2]$ is zero-mean Gaussian noise, independent from one trial to the next.[1] For the cases above, $\mathbf{x}_n = (x_n^L, x_n^S)$ might have two dimensions, one each for light and sound, taking on values that are binary, representing the presence and absence of the stimuli. Similarly, $\mathbf{w}_n = (w_n^L, w_n^S)$ also has two dimensions. Crucially, to allow for the possibility (realized in most conditioning experiments) that the true weights might change, the model includes a diffusion term

$$\mathbf{w}_{n+1} = \mathbf{w}_n + \boldsymbol{\eta}_n \tag{2}$$

where $\boldsymbol{\eta}_n \sim N[0, \sigma^2 \mathbb{I}]$ is also Gaussian. The task for the animal is to take observations of the stimuli $\{\mathbf{x}_n\}$ and rewards $\{r_n\}$ and *infer* a distribution over $\mathbf{w}_n$. Provided that the initial uncertainty can be captured as $\mathbf{w}_0 \sim N[0, \Sigma_0]$ for some covariance matrix $\Sigma_0$, inference takes the form of a standard recursive Kalman filter, for which $\mathcal{P}(\mathbf{w}_n | r_1 ... r_{n-1}) \sim N[\hat{\mathbf{w}}_n, \Sigma_n]$ and

$$\hat{\mathbf{w}}_{n+1} = \hat{\mathbf{w}}_n + \frac{\Sigma_n \cdot \mathbf{x}_n}{\mathbf{x}_n \cdot \Sigma_n \cdot \mathbf{x}_n + \tau^2} (r_n - \hat{\mathbf{w}}_n \cdot \mathbf{x}_n) \tag{3}$$

$$\Sigma_{n+1} = \Sigma_n + \sigma^2 \mathbb{I} - \frac{\Sigma_n \cdot \mathbf{x}_n \mathbf{x}_n \cdot \Sigma_n}{\mathbf{x}_n \cdot \Sigma_n \cdot \mathbf{x}_n + \tau^2} \tag{4}$$

If $\Sigma_n \propto \mathbb{I}$, then the update for the mean can be seen as a standard delta rule (Widrow & Stearns, 1985; Rescorla & Wagner, 1972), involving the prediction error (or innovation) $\delta_n = r_n - \hat{\mathbf{w}}_n \cdot \mathbf{x}_n$. Note the familiar, but at first sight counterintuitive, result that the update for the covariance matrix does not depend on the innovation or the observed $r_n$.[2]

In backward blocking, in the first set of trials, the off-diagonal terms of the covariance matrix $\Sigma_n$ become *negative*. This can either be seen from the form of the update equation for the covariance matrix (since $\mathbf{x}_n \sim (1,1)$), or, more intuitively, from the fact that these trials imply a constraint only on $w_n^L + w_n^S$, therefore forcing $w_n^S$ and $w_n^L$ to be negatively correlated. The consequence of this negative correlation in the second set of trials is that the S component of $\Sigma_n \cdot \mathbf{x}_n = \Sigma_n \cdot (1,0)$ is less than 0, and so, via equation 3, $\hat{w}_n^S$ reduces. This is exactly the result in backward blocking. Another way of looking at this is in terms of explaining away in weight space. From the first set of trials, the animal infers that $w_n^L + w_n^S = R > 0$; from the second, that the prediction owes to $w_n^L$ rather than $w_n^S$, and so the old value $w_n^S = R/2$ is explained away by $w_n^L$. Sutton (1992) actually suggested the approximation of forcing the off-diagonal components of the covariance matrix $\Sigma_n$ to be 0, which, of course, prevents the system from accounting for backward blocking.

We seek a network account of explaining away in the space of weights by implementing an approximate form of Kalman filtering.

## 3   Whitening and the Kalman filter

In conventional applications of the Kalman filter, $\mathbf{x}_n$ would typically be *constant*. That is, the hidden state ($\mathbf{w}_n$) would be observed through a fixed observation process. In cases such as classical conditioning, though, this is not true – we are interested in the case that $\mathbf{x}_n$ changes over time, possibly even in a random (though fully observable) way. The plan for this section is to derive an approximate relationship between the average covariance matrix over the weights $\bar{\Sigma}$ and a whitening matrix for the stimulus inputs. In the next section, we consider an implementation of a particular whitening algorithm as an unsupervised way of estimating the covariance matrix for the Kalman filter and show how to use it to learn the weights $\hat{\mathbf{w}}_n$ appropriately.

Consider the case that $\mathbf{x}_n$ are random, with correlation matrix $\langle \mathbf{xx} \rangle = Q$, and consider the *mean* covariance matrix $\bar{\Sigma}$ for the Kalman filter, averaging across the variation in $\mathbf{x}$. Make the approximation that

$$\left\langle \frac{\bar{\Sigma} \cdot \mathbf{xx} \cdot \bar{\Sigma}}{\mathbf{x} \cdot \bar{\Sigma} \cdot \mathbf{x} + \tau^2} \right\rangle = \frac{\langle \bar{\Sigma} \cdot \mathbf{xx} \cdot \bar{\Sigma} \rangle}{\langle \mathbf{x} \cdot \bar{\Sigma} \cdot \mathbf{x} + \tau^2 \rangle}$$

which is less drastic than it might first appear since the denominator is just a scalar. Then, we can solve for the average of the asymptotic value of $\bar{\Sigma}$ in the equation for the update of the Kalman filter as

$$\bar{\Sigma} Q \bar{\Sigma} \propto \mathbb{I} \tag{5}$$

Thus $\bar{\Sigma}$ is a whitening filter for the correlation matrix $Q$ of the inputs $\{\mathbf{x}\}$. Symmetric whitening filters ($\bar{\Sigma}$ must be symmetric) are generally unique (Atick & Redlich, 1993). This result is very different from the standard relationship between Kalman filtering and whitening. The standard Kalman filter is a whitening filter for the innovations process $\delta_n = r_n - \hat{\mathbf{w}}_n \cdot \mathbf{x}_n$, *ie* as extracting all the systematic variation into $\mathbf{w}_n$, leaving only random variation due to the observation noise and the diffusion process. Equation 5 is an additional level of whitening, saying that one can look at the long-run average covariance

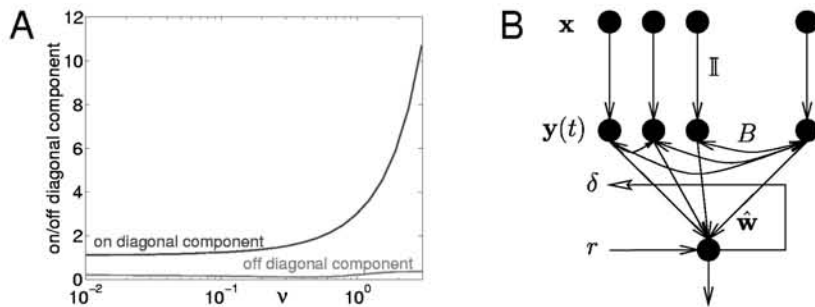

Figure 1: Whitening. A) The lower curve shows the average maximum off-diagonal element of $|\bar{\Sigma}Q\bar{\Sigma}|$ as a function of $v$. The upper curve shows the average maximum diagonal element of the same matrix. The off-diagonal components are around an order of magnitude smaller than the on-diagonal components, even in the difficult regime where $v$ is near 0, and thus the matrix $Q$ is nearly singular. B) Network model for Kalman filtering. Identity feedforward weights $\mathbb{I}$ map inputs $\mathbf{x}$ to a recurrent network $\mathbf{y}(t)$ whose output is used to make predictions. Learning of the recurrent weights $B$ is based on Goodall's (1960) rule; learning of the prediction weights is based on the delta rule, only using $\mathbf{y}(0)$ to make the predictions and $\mathbf{y}(\infty)$ to change the weights.

matrix of the uncertainty in $\mathbf{w}_n$ as whitening the input process $\mathbf{x}_n$. This is inherently unsupervised, in that whitening takes place without any reference to the observed rewards (or even the innovation).

Given the approximation, we tested whether $\bar{\Sigma}$ really whitens $Q$ by by generating $\mathbf{x}_n$ from a Gaussian distribution, with mean $(1, 1)$ and variance $v^2\mathbb{I}$, calculating the long-run average value of $\bar{\Sigma}$, and assessing whether $\Gamma = \bar{\Sigma}Q\bar{\Sigma}$ is white. There is no unique measure for the deviation of $\Gamma$ from being diagonal; as an example, figure 1A shows as a function of $v$ the largest on- and off-diagonal elements of $\Gamma$. The figure shows that the off-diagonal components are comparatively very small, even when $v$ is very small, for which $Q$ has an eigenvalue very near to 0 making the whitening matrix nearly undefined. Equally, in this case, $\Sigma_n$ tends to have very large values, since, looking at equation 4, the growth in uncertainty coming from $\sigma^2\mathbb{I}$ is not balanced by any observation in the direction $(1, -1)$ that is orthogonal to $(1, 1)$.

Of course, only the long-run average covariance matrix $\bar{\Sigma}$ whitens $Q$. We make the further approximation of using an on-line estimate of the symmetric whitening matrix as the on-line estimate of the covariance of the weights $\Sigma_n$.

## 4   A network model

Figure 1B shows a network model in which prediction weights $\hat{\mathbf{w}}_n$ adapt in a manner that is appropriately sensitive to a learned, on-line, estimate of the whitening matrix. The network has two components, a mapping from input $\mathbf{x}$ to output $\mathbf{y}(t)$, via recurrent feedback weights $B$ (the Goodall (1960) whitening filter), and a mapping from $\mathbf{y}$, through a set of prediction weights $\hat{\mathbf{w}}$ to an estimate of the reward. The second part of the network is most straightforward. The feedforward weights from $\mathbf{x}$ to $\mathbf{y}$ are just the identity matrix $\mathbb{I}$. Therefore, the initial value in the hidden layer in response to stimulus $\mathbf{x}_n$ is $\mathbf{y}(0) = \mathbf{x}_n$, and so the prediction of reward is just $\hat{\mathbf{w}} \cdot \mathbf{y}(0) = \hat{\mathbf{w}} \cdot \mathbf{x}_n$.

The first part of the network is a straightforward implementation of Goodall's whitening filter (Goodall, 1960; Atick & Redlich, 1993). The recurrent dynamics in the $\mathbf{y}$-layer are taken as being purely linear. Therefore, in response to input $\mathbf{x}$ (propagated through the

identity feedforward weights)

$$\tau \dot{\mathbf{y}} = -\mathbf{y} + \mathbf{x} + B\mathbf{y}$$

and so $\mathbf{y}(\infty) = (\mathbb{I} - B)^{-1}\mathbf{x}$, provided that the inverse exists. Goodall's algorithm changes the recurrent weights $B$ using local, anti-Hebbian learning, according to

$$\Delta B \propto -\mathbf{x}\mathbf{y} + \mathbb{I} - B . \qquad (6)$$

This rule stabilizes on average when $\mathbb{I} = (\mathbb{I} - B)^{-1}Q[(\mathbb{I} - B)^{-1}]$, that is when $(\mathbb{I} - B)^{-1}$ is a whitening filter for the correlation matrix $Q$ of the inputs. If $B$ is symmetric, which can be guaranteed by making $B = \mathbb{O}$ initially (Atick & Redlich, 1993), then, by convergence, we have $(\mathbb{I} - B)^{-1} = \bar{\Sigma}$ and, given input $\mathbf{x}_n$ to the network

$$\bar{\Sigma}\mathbf{x}_n = (\mathbb{I} - B)^{-1}\mathbf{x}_n = \mathbf{y}_n(\infty)$$

Therefore, we can implement a learning rule for the prediction weights akin to the Kalman filter (equation 3) using

$$\Delta \hat{\mathbf{w}}_n \propto \mathbf{y}_n(\infty)\,(r_n - \hat{\mathbf{w}}_n \cdot \mathbf{y}_n(0)) . \qquad (7)$$

This is the standard delta rule, except that the predictions are based on $\mathbf{y}_n(0) = \mathbf{x}_n$, whereas the weight changes are based on $\mathbf{y}_n(\infty) = \bar{\Sigma}\mathbf{x}_n$. The learning rule gets wrong the absolute magnitude of the weight changes (since it lacks the $\mathbf{x}_n \cdot \Sigma_n \cdot \mathbf{x}_n + \tau^2$ term on the denominator – but it gets right the direction of the changes.

## 5  Results

Figure 2 shows the result of learning in backward blocking. In association with $r_n = 1$, first stimulus $\mathbf{x}_n = (1,1)$ was presented for 20 trials, then stimulus $\mathbf{x}_n = (1,0)$ was presented for a further 20 trials. Figure 2A shows the development of the weights $\hat{w}_n^L$ (solid) and $\hat{w}_n^S$ (dashed). During the first set of trials, these grow towards 0.5; during the second set, they differentiate sharply with the weight associated with the light growing towards 1, and that with the sound, which is explained away, growing towards 0. Figure 2B shows the development of two terms in the estimated covariance matrix. The negative covariance between light and sound is evident, and causes the sharp changes in the weights on the 21st trial. Figure 2C & D show the values using the exact Kalman filter, showing qualitatively similar behavior.

The increases in the magnitudes of $\Sigma_n^{LL}$ and $\Sigma_n^{LS}$ during the first stage of backwards blocking come from the lack of information in the input about $w_n^L - w_n^S$, despite its continual diffusion (from equation 2). Thus backwards blocking is a pathological case. Nevertheless, the on-line method for estimating $\Sigma$ captures the correct behavior. Figures 2 E-H show a non-pathological case with observation noise added. The estimates from the model closely match those of the exact Kalman filter, a result that is is also true for other non-pathological cases.

## 6  Discussion

We have shown how the standard Kalman filter produces explaining away in the space of weights, and suggested and proved efficacious a natural network model for implementing the Kalman filter. The model mixes unsupervised learning of a whitener for the observation process (*ie* the $\mathbf{x}_n$ of equation 1), providing the covariance matrix governing the uncertainty in the weights, with supervised (or equivalently reinforcement) learning of the mean values of the weights. Unsupervised learning is reasonable since the evolution of the covariance matrix of the weights is independent of the innovations. The basic result is an

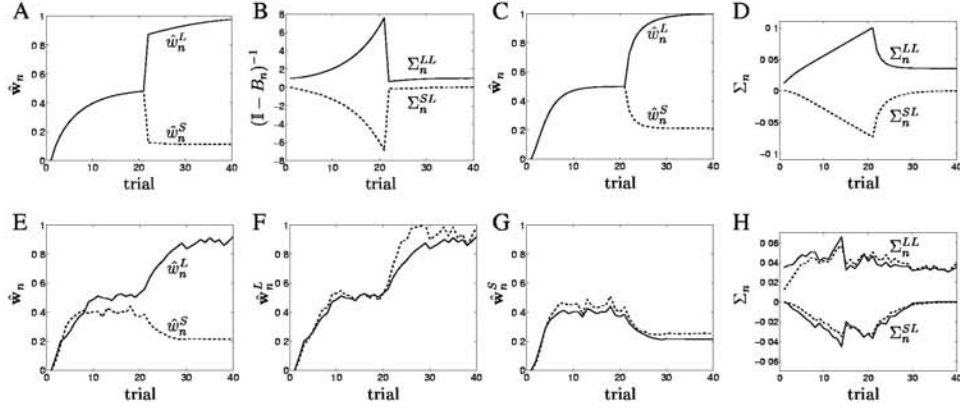

Figure 2: Backward blocking in the full model. A) The development of $\hat{\mathbf{w}}$ over 20 trials with $\mathbf{x}_n = (1,1)$ and 20 with $\mathbf{x}_n = (1,0)$. B) The development of the estimated covariance of the weight for the light $\Sigma_n^{LL}$ and cross-covariance between the light and the sound $\Sigma_n^{LS}$. The learning rates in equations 6 and 7 were both 0.125. C & D) The development of $\hat{\omega}$ and $\Sigma$ from the exact Kalman filter with parameters ($\sigma = .09$ and $\tau = 0.35$). E) The development of $\hat{\mathbf{w}}$ as in A) except with multiplicative Gaussian noise added (*ie* noise with standard deviation 0.35 is added only to the representations of stimuli that are present). F & G) The comparison of $\hat{\mathbf{w}}$ in the model (solid line) and in the exact Kalman filter (dashed line), using the same parameters for the Kalman filter as in C) and D). H) A comparison of the true covariance, $\Sigma_n$ (dashed line), with the rescaled estimate, $(\mathbb{I} - B)^{-1}$ (solid line).

approximation, but one that has been shown to match results quite closely. Further work is needed to understand how to set the parameters of the Goodall learning rule to match $\sigma^2$ and $\tau^2$ exactly.

Hinton (personal communication) has suggested an alternative interpretation of Kalman filtering based on a heteroassociative novelty filter. Here, the idea is to use the recurrent network $B$ only once, rather than to equilibrium, with (as for our model) $\mathbf{y}_n(0) = \mathbf{x}_n$, the prediction $v = \hat{\mathbf{w}}_n \cdot \mathbf{y}_n(0)$, $\mathbf{y}_n(1) = B_n \cdot \mathbf{x}_n$, and

$$\Delta \hat{\mathbf{w}}_n \propto \mathbf{y}_n(1) (r_n - \hat{\mathbf{w}}_n \cdot \mathbf{y}_n(0)) \ .$$

This gives $B_n$ a similar role to $\Sigma_n$ in learning $\hat{\mathbf{w}}_n$. For the novelty filter,

$$\Delta B_n = -\frac{B_n \cdot \mathbf{x}_n \mathbf{x}_n \cdot B_n}{|B_n \cdot \mathbf{x}_n|^2} \ ,$$

which makes the network a perfect heteroassociator between $\mathbf{x}_n$ and $r_n$. If we compare the update for $B_n$ to that for $\Sigma_n$ (equation 4), we can see that it amounts approximately to assuming neither observation noise nor drift. Thus, whereas our network model approximates the long-run covariance matrix, the novelty filter approximates the instantaneous covariance matrix directly, and could clearly be adapted to take account of noise. Unfortunately, there are few quantitatively precise experimental results on backwards blocking, so it is hard to choose between different possible rules.

There is a further alternative. Sutton (1992) suggested an online way of estimating the elements of the covariance matrix, observing that

$$E[\delta_n^2] = \tau^2 + \mathbf{x}_n \cdot \Sigma_n \cdot \mathbf{x}_n \tag{8}$$

and so considered using a standard delta rule to fit the square innovation using a quadratic input representation $\left((x_n^L)^2, (x_n^S)^2, x_n^L \times x_n^S, 1\right)$.[3] The weight associated with the last ele-

ment, *ie* the bias, should come to be the observation noise $\tau^2$; the weights associated with the other elements are just the components of $\Sigma_n$. The most critical concern about this is that it is not obvious how to use the resulting covariance matrix to control learning about the mean values of the weights. There is also the more theoretical concern that the covariance matrix should really be independent of the prediction errors, one manifestation of which is that the occurrence of backward blocking in the model of equation 8 is strongly sensitive to initial conditions.

Although backward blocking is a robust phenomenon, particularly in human conditioning experiments (Shanks, 1985), it is not observed in all animal conditioning paradigms. One possibility for why not is that the anatomical substrate of the cross-modal recurrent network (the $B$ weights in the model) is not ubiquitously available. In its absence, $\mathbf{y}(\infty) = \mathbf{y}(0) = \mathbf{x}_n$ in response to an input $\mathbf{x}_n$, and so the network will perform like the standard delta or Rescorla-Wagner (Rescorla & Wagner, 1972) rule.

The Kalman filter is only one part of a more complicated picture for statistically normative models of conditioning. It makes for a particularly clear example of what is incomplete about some of our own learning rules (notably Kakade & Dayan, 2000) which suggest that, at least in some circumstances, learning about the two different stimuli should progress completely independently. We are presently trying to integrate on-line and learned competitive and additive effects using ideas from mixture models and Kalman filters.

## Acknowledgements

We are very grateful to David Shanks, Rich Sutton, Read Montague and Terry Sejnowski for discussions of the Kalman filter model and its relationship to backward blocking, and to Sam Roweis for comments on the paper. This work was funded by the Gatsby Charitable Foundation and the NSF.

## Footnotes

[1] For vectors $\mathbf{a}, \mathbf{b}$, matrix $\mathbf{C}$, $\mathbf{a} \cdot \mathbf{b} = \sum_i a_i b_i$, $\mathbf{a} \cdot \mathbf{C} \cdot \mathbf{b} = \sum_{ij} a_i C_{ij} b_j$, matrix $[\mathbf{ab}]_{ij} = a_i b_j$.

[2]Note also the use of the alternative form of the Kalman filter, in which we perform observation/conditioning followed by drift, rather than drift followed by observation/conditioning.

[3]Although the $x_n^L \times x_n^S$ term was omitted from Sutton's diagonal approximation to $\Sigma_n$.

## References

Atick, JJ & Redlich, AN (1993) Convergent algorithm for sensory receptive field development. *Neural Computation* **5**:45-60.

Goodall, MC (1960) Performance of stochastic net. *Nature* **185**:557-558.

Jordan, MI, editor (1998) *Learning in Graphical Models.* Dordrecht: Kluwer.

Kakade, S & Dayan, P (2000) Acquisition in autoshaping. In SA Solla, TK Leen & K-R Muller, editors, *Advances in Neural Information Processing Systems, 12.*

Miller, RR & Matute, H (1996). Biological significance in forward and backward blocking: Resolution of a discrepancy between animal conditioning and human causal judgment. *Journal of Experimental Psychology: General* **125**:370-386.

Rescorla, RA & Wagner, AR (1972) A theory of Pavlovian conditioning: The effectiveness of reinforcement and non-reinforcement. In AH Black & WF Prokasy, editors, *Classical Conditioning II: Current Research and Theory.* New York:Aleton-Century-Crofts, 64-69.

Shanks, DR (1985). Forward and backward blocking in human contingency judgement. *Quarterly Journal of Experimental Psychology: Comparative & Physiological Psychology* **37**:1-21.

Sutton, RS (1992). Gain adaptation beats least squares? In *Proceedings of the 7th Yale Workshop on Adaptive and Learning Systems.*

Wagner, AR & Brandon, SE (1989). Evolution of a structured connectionist model of Pavlovian conditioning (AESOP). In SB Klein & RR Mowrer, editors, *Contemporary Learning Theories.* Hillsdale, NJ: Erlbaum, 149-189.

Widrow, B & Stearns, SD (1985) *Adaptive Signal Processing.* Englewood Cliffs, NJ:Prentice-Hall.
